# Efficient Moments-based Permutation Tests

**Chunxiao Zhou**
Dept. of Electrical and Computer Eng.
University of Illinois at Urbana-Champaign
Champaign, IL 61820
*czhou4@gmail.com*

**Huixia Judy Wang**
Dept. of Statistics
North Carolina State University
Raleigh, NC 27695
*wang@stat.ncsu.edu*

**Yongmei Michelle Wang**
Depts. of Statistics, Psychology, and Bioengineering
University of Illinois at Urbana-Champaign
Champaign, IL 61820
*ymw@illinois.edu*

## Abstract

In this paper, we develop an efficient moments-based permutation test approach to improve the test's computational efficiency by approximating the permutation distribution of the test statistic with Pearson distribution series. This approach involves the calculation of the first four moments of the permutation distribution. We propose a novel recursive method to derive these moments theoretically and analytically without any permutation. Experimental results using different test statistics are demonstrated using simulated data and real data. The proposed strategy takes advantage of nonparametric permutation tests and parametric Pearson distribution approximation to achieve both accuracy and efficiency.

## 1    Introduction

Permutation tests are flexible nonparametric alternatives to parametric tests in small samples, or when the distribution of a test statistic is unknown or mathematically intractable. In permutation tests, except exchangeability, no other statistical assumptions are required. The *p*-values can be obtained by using the permutation distribution. Permutation tests are appealing in many biomedical studies, which often have limited observations with unknown distribution. They have been used successfully in structural MR image analysis [1, 2, 3], in functional MR image analysis [4], and in 3D face analysis [5].

There are three common approaches to construct the permutation distribution [6, 7, 8]: (1) exact permutation enumerating all possible arrangements; (2) approximate permutation based on random sampling from all possible permutations; (3) approximate permutation using the analytical moments of the exact permutation distribution under the null hypothesis. The main disadvantage of the exact permutation is the computational cost, due to the factorial increase in the number of permutations with the increasing number of subjects. The second technique often gives inflated type I errors caused by random sampling. When a large number of repeated tests are needed, the random permutation strategy is also computationally expensive to achieve satisfactory accuracy. Regarding the third approach, the exact permutation distribution may not have moments or moments with tractability. In most applications, it is not the existence but the derivation of moments that limits the third approach.

To the best of our knowledge, there is no systematic and efficient way to derive the moments of the permutation distribution. Recently, Zhou [3] proposed a solution by converting the permutation of data to that of the statistic coefficients that are symmetric to the permutation. Since the test statistic coefficients usually have simple presentations, it is easier to track the permutation of the test statistic coefficients than that of data. However, this method requires the derivation of the permutation for each specific test statistic, which is not accessible to practical users.

In this paper, we propose a novel strategy by employing a general theoretical method to derive the moments of the permutation distribution of any weighted $v$-statistics, for both univariate and multivariate data. We note that any moments of the permutation distribution for weighted $v$-statistics [9] can be considered as a summation of the product of data function term and index function term over a high dimensional index set and all possible permutations. Our key idea is to divide the whole index set into several permutation equivalent (see Definition 2) index subsets such that the summation of the data/index function term over all permutations is invariant within each subset and can be calculated without conducting any permutation. Then we can obtain the moments by summing up several subtotals. The proposed method can be extended to equivalent weighted $v$-statistics by replacing them with monotonic weighted $v$-statistics. This is due to the fact that only the order of test statistics of all permutations matters for obtaining the $p$-values, so that the monotonic weighted $v$-statistics shares the same $p$-value with the original test statistic. Given the first four moments, the permutation distribution can be well fitted by Pearson distribution series. The $p$-values are then obtained without conducting any real permutation. For multiple comparison of two-group difference, given the sample size $n_1 = 21$ and $n_2 = 21$, the number of tests $m = 2,000$, we need to conduct $m \times (n_1 + n_2)!/n_1!/n_2! \approx 1.1 \times 10^{15}$ permutations for the exact permutation test. Even for 20,000 random permutations per test, we still need $m \times 20,000 \approx 4 \times 10^7$ permutations. Alternatively, our moments-based permutation method using Pearson distribution approximation only involves the calculation of the first four analytically-derived moments of exact permutation distributions to achieve high accuracy (see section 3). Instead of calculating test statistics in factorial scale with exact permutation, our moments-based permutation only requires computation of polynomial order. For example, the computational cost for univariate mean difference test statistic and modified multivariate Hotelling's $T^2$ test statistics [8] are $O(n)$ and $O(n^3)$, respectively, where $n = n_1 + n_2$.

## 2    Methodology

In this section, we shall mainly discuss how to calculate the moments of the permutation distribution for weighted $v$-statistics.  For other test statistics, a possible solution is to replace them with their equivalent weighted $v$-statistics by monotonic transforms. The detailed discussion about equivalent test statistics can be found in [7, 8, 10].

### 2.1    Computational challenge

Let us first look at a toy example. Suppose we have a two-group univariate data $x = (x_1, \cdots, x_{n_1}, x_{n_1+1}, \cdots, x_{n_1+n_2})$, where the first $n_1$ elements are in group A and the rest, $n_2$, are in group B. For comparison of the two groups, the hypothesis is typically constructed as: $H_0: \mu_A = \mu_B$ vs. $H_a: \mu_A \neq \mu_B$, where $\mu_A, \mu_B$ are the population means of the groups A and B, respectively. Define $\bar{x}_A = \sum_{i=1}^{n_1} x_i / n_1$ and $\bar{x}_B = \sum_{i=n_1+1}^{n} x_i / n_2$ as the sample means of two groups, where $n=n_1+n_2$. We choose the univariate group mean difference  as the test statistic, i.e., $T(x) = \bar{x}_A - \bar{x}_B = \sum_{i=1}^{n} w(i)x_i$, where the index function $w(i) = 1/n_1$, if $i \in \{1, \cdots, n_1\}$ and $w(i) = -1/n_2$, if $i \in \{n_1+1, \cdots, n\}$. Then the total number of all possible permutations of $\{1, \cdots, n\}$ is $n!$. To calculate the fourth moment of the permutation distribution,

$$E_\pi(T^4(x)) = \frac{1}{n!}\sum_{\pi\in S_n}(\sum_{i=1}^n w(i)x_{\pi(i)})^4 = \frac{1}{n!}\sum_{\pi\in S_n}\sum_{i_1=1}^n\sum_{i_2=1}^n\sum_{i_3=1}^n\sum_{i_4=1}^n w(i_1)w(i_2)w(i_3)w(i_4)x_{\pi(i_1)}x_{\pi(i_2)}x_{\pi(i_3)}x_{\pi(i_4)},$$

where $\pi$ is the permutation operator and the symmetric group $S_n$ [11] includes all distinct permutations. The above example shows that the moment calculation can be considered as a summation over all possible permutations and a large index set. It is noticeable that the computational challenge here is to go through the factorial level permutations and polynomial level indices.

## 2.2    Partition the index set

In this paper, we assume that the test statistic $T$ can be expressed as a weighted $v$-statistic of degree $d$ [9], that is, $T(x) = \sum_{i_1=1}^n\cdots\sum_{i_d=1}^n w(i_1,\cdots,i_d)h(x_{i_1},\cdots,x_{i_d})$, where $x = (x_1, x_2,\cdots,x_n)^T$ is a data with $n$ observations, and $w$ is a symmetric index function. $h$ is a symmetric data function, i.e., invariant under permutation of $(i_1,\cdots,i_d)$. Though the symmetry property is not required for our method, it helps reduce the computational cost. Here, each observation $x_k$ can be either univariate or multivariate. In the above toy example, $d=1$ and $h$ is the identity function. Therefore, the $r$-th moment of the test statistic from the permutated data is:

$$E_\pi(T^r(x)) = E_\pi(\sum_{i_1,i_2,\cdots,i_d} w(i_1,\cdots,i_d)h(x_{\pi(i_1)},\cdots,x_{\pi(i_d)}))^r$$

$$= E_\pi[\sum_{\substack{i_1^{(1)},\cdots,i_d^{(1)},\\ \cdots\\ i_1^{(r)},\cdots,i_d^{(r)}}}\{\prod_{k=1}^r w(i_1^{(k)},\cdots,i_d^{(k)})\prod_{k=1}^r h(x_{\pi(i_1^{(k)})},\cdots,x_{\pi(i_d^{(k)})})\}].$$

Then we can exchange the summation order of permutations and that of indices,

$$E_\pi(T^r(x)) = \sum_{\substack{i_1^{(1)},\cdots,i_d^{(1)},\\ \cdots\\ i_1^{(r)},\cdots,i_d^{(r)}}}\{(\prod_{k=1}^r w(i_1^{(k)},\cdots,i_d^{(k)}))E_\pi(\prod_{k=1}^r h(x_{\pi(i_1^{(k)})},\cdots,x_{\pi(i_d^{(k)})}))\}.$$

Thus any moment of permutation distribution can be considered as a summation of the product of data function term and index function term over a high dimensional index set and all possible permutations.

Since all possible permutations map any index value between 1 and $n$ to all possible index values from 1 to $n$ with equal probability, $E_\pi(\prod_{k=1}^r h(x_{\pi(i_1^{(k)})},\cdots,x_{\pi(i_d^{(k)})}))$, the summation of data function over all permutations is only related to the equal/unequal relationship among indices. It is natural to divide the whole index set $U = \{i_1,\cdots,i_d\}^r = \{(i_1^{(1)},\cdots,i_d^{(1)}),\cdots,(i_1^{(r)},\cdots,i_d^{(r)})\}$ into the union of disjoint index subsets, in which $E_\pi(\prod_{k=1}^r h(x_{\pi(i_1^{(k)})},\cdots,x_{\pi(i_d^{(k)})}))$ is invariant.

**Definition 1**. Since $h$ is a symmetric function, two index elements $(i_1,\cdots,i_d)$ and $(j_1,\cdots,j_d)$ are said to be equivalent if they are the same up to the order. For example, for $d = 3$, (1, 4, 5) = (1,5,4) = (4,1,5) = (4,5,1) = (5,1,4) = (5,4,1).

**Definition 2**. Two indices $\{(i_1^{(1)},\cdots,i_d^{(1)}),\cdots,(i_1^{(r)},\cdots,i_d^{(r)})\}$ and $\{(j_1^{(1)},\cdots,j_d^{(1)}),\cdots,(j_1^{(r)},\cdots,j_d^{(r)})\}$ are said to be permutation equivalent/ if there exists a permutation $\pi \in S_n$ such that $\{(\pi(i_1^{(1)}),\cdots,\pi(i_d^{(1)})),\cdots,(\pi(i_1^{(r)}),\cdots,\pi(i_d^{(r)}))\} = \{(j_1^{(1)},\cdots,j_d^{(1)}),\cdots,(j_1^{(r)},\cdots,j_d^{(r)})\}$. Here "=" means they have same index elements by Definition 1. For example, for $d = 2$, $n = 4$, $r = 2$, $\{(1, 2), (2, 3)\}$ $\{(2, 4), (1, 4)\}$ since we can apply $\pi$: 1 1, 2 4, 3 2, 4 3, such that $\{(\pi(1), \pi(2)), (\pi(2), \pi(3))\} = \{(1, 4), (4, 2)\} = \{(2, 4), (1, 4)\}$. As a result, the whole index set for $d = 2$, $r = 2$, can be divided into seven permutation equivalent subsets, [{(1, 1), (1, 1)}], [{(1, 1), (1, 2)}], [{(1, 1), (2, 2)}], [{(1, 2), (1, 2)}], [{(1, 1), (2, 3)}], [{(1, 2), (1, 3)}], [{(1, 2), (3, 4)}], where [ ] denotes the equivalence class. Note that the number of the permutation equivalent subsets is only related to the order of weighted $v$-test statistic $d$ and the order of moment $r$,

but not related to the data size $n$, and it is small for the first several moments calculation (small $r$) with low order test statistics (small $d$).

Using the permutation equivalent relationship defined in Definition 2, the whole index set $U$ can be partitioned into several permutation equivalent index subsets. Then we can calculate the $r$-th moment by summing up subtotals of all index subsets. This procedure can be done without any real permutations based on Proposition 1 and Proposition 2 below.

**Proposition 1**. We claim that the data function sum $E_\pi(\prod_{k=1}^{r} h(x_{\pi(i_1^{(k)})}, \cdots, x_{\pi(i_d^{(k)})}))$ is invariant within each equivalent index subset, and

$$E_\pi(\prod_{k=1}^{r} h(x_{\pi(i_1^{(k)})}, \cdots, x_{\pi(i_d^{(k)})})) = \frac{\sum\limits_{\{(j_1^{(1)}, \cdots, j_d^{(1)}), \cdots, (j_1^{(r)}, \cdots, j_d^{(r)})\} \in [\{(i_1^{(1)}, \cdots, i_d^{(1)}), \cdots, (i_1^{(r)}, \cdots, i_d^{(r)})\}]} \prod_{k=1}^{r} h(x_{j_1^{(k)}}, \cdots, x_{j_d^{(k)}})}{card([\{(i_1^{(1)}, \cdots, i_d^{(1)}), \cdots, (i_1^{(r)}, \cdots, i_d^{(r)})\}])},$$

where $card([\{(i_1^{(1)}, \cdots, i_d^{(1)}), \cdots, (i_1^{(r)}, \cdots, i_d^{(r)})\}])$ is the number of indices falling into the permutation equivalent index subset $[\{(i_1^{(1)}, \cdots, i_d^{(1)}), \cdots, (i_1^{(r)}, \cdots, i_d^{(r)})\}]$.

**Proof sketch**:

Since all indices in the same permutation equivalent subset are equivalent with respect to the symmetric group $S_n$,

$$E_\pi(\prod_{k=1}^{r} h(x_{\pi(i_1^{(k)})}, \cdots, x_{\pi(i_d^{(k)})})) = \frac{1}{n!} \sum_{\pi \in S_n} \prod_{k=1}^{r} h(x_{\pi(i_1^{(k)})}, \cdots, x_{\pi(i_d^{(k)})}) =$$

$$= \frac{\dfrac{\sum\limits_{\{(j_1^{(1)}, \cdots, j_d^{(1)}), \cdots, (j_1^{(r)}, \cdots, j_d^{(r)})\} \in [\{(i_1^{(1)}, \cdots i_d^{(1)}), \cdots, (i_1^{(r)}, \cdots i_d^{(r)})\}]} (\prod_{k=1}^{r} h(x_{j_1^{(k)}}, \cdots, x_{j_d^{(k)}}) \, n!)}{card([\{(i_1^{(1)}, \cdots, i_d^{(1)}), \cdots, (i_1^{(r)}, \cdots, i_d^{(r)})\}])}}{n!},$$

$$= \frac{\sum\limits_{\{(j_1^{(1)}, \cdots, j_d^{(1)}), \cdots, (j_1^{(r)}, \cdots, j_d^{(r)})\} \in [\{(i_1^{(1)}, \cdots i_d^{(1)}), \cdots, (i_1^{(r)}, \cdots i_d^{(r)})\}]} (\prod_{k=1}^{r} h(x_{j_1^{(k)}}, \cdots, x_{j_d^{(k)}}))}{card([\{(i_1^{(1)}, \cdots, i_d^{(1)}), \cdots, (i_1^{(r)}, \cdots, i_d^{(r)})\}])}.$$

**Proposition 2**. Thus we can obtain the $r$-th moment by summing up the production of the data partition sum $w_\lambda$ and the index partition sum $h_\lambda$ over all permutation equivalent subsets, i.e.,

$$E_\pi(T^r(x)) = \sum_{\lambda \in [U]} w_\lambda h_\lambda, \quad \text{where} \quad \lambda = [\{(i_1^{(1)}, \cdots, i_d^{(1)}), \cdots, (i_1^{(r)}, \cdots, i_d^{(r)})\}] \text{ is any permutation}$$

equivalent subset of the whole index set $U$. $[U]$ denotes the set of all distinct permutation equivalent classes of $U$. The data partition sum is

$$h_\lambda = \frac{\sum\limits_{\{(j_1^{(1)}, \cdots, j_d^{(1)}), \cdots, (j_1^{(r)}, \cdots, j_d^{(r)})\} \in \lambda} (\prod_{k=1}^{r} h(x_{j_1^{(k)}}, \cdots, x_{j_d^{(k)}}))}{card(\lambda)}, \quad \text{and the index partition sum is}$$

$$w_\lambda = \sum_{\{(j_1^{(1)}, \cdots, j_d^{(1)}), \cdots, (j_1^{(r)}, \cdots, j_d^{(r)})\} \in \lambda} (\prod_{k=1}^{r} w(x_{j_1^{(k)}}, \cdots, x_{j_d^{(k)}})).$$

**Proof sketch**:

With Proposition 1, $E_\pi(\prod_{k=1}^{r} h(x_{\pi(i_1^{(k)})}, \cdots, x_{\pi(i_d^{(k)})}))$ is invariant within each equivalent index subset, therefore,

$$E_\pi(T^r(x)) = \sum_{\substack{i_1^{(1)}, \cdots, i_d^{(1)}, \\ \cdots \\ i_1^{(r)}, \cdots, i_d^{(r)}}} \{(\prod_{k=1}^{r} w(i_1^{(k)}, \cdots, i_d^{(k)})) E_\pi(\prod_{k=1}^{r} h(x_{\pi(i_1^{(k)})}, \cdots, x_{\pi(i_d^{(k)})}))\} =$$

$$= \sum_{\lambda \in [U]} \sum_{\{(j_1^{(1)},\cdots,j_d^{(1)}),\cdots,(j_1^{(r)},\cdots,j_d^{(r)})\} \in \lambda} \{(\prod_{k=1}^{r} w(j_1^{(k)},\cdots,j_d^{(k)})) E_\pi (\prod_{k=1}^{r} h(x_{\pi(j_1^{(k)})},\cdots,x_{\pi(j_d^{(k)})}))\} =$$

$$= \sum_{\lambda \in [U]} \sum_{\{(j_1^{(1)},\cdots,j_d^{(1)}),\cdots,(j_1^{(r)},\cdots,j_d^{(r)})\} \in \lambda} \{\prod_{k=1}^{r} w(j_1^{(k)},\cdots,j_d^{(k)}) h_\lambda\} = \sum_{\lambda \in [U]} w_\lambda h_\lambda .$$

Since both data partition sum $w_\lambda$ and the index partition sum $h_\lambda$ can be calculated by summation over all distinct indices within each permutation equivalent index subset, no any real permutation is needed for computing the moments.

## 2.3 Recursive calculation

Direct calculation of the data partition sum and index partition sum leads to traversing throughout the whole index set. So the computational cost is $O(n^{\mathrm{dr}})$. In the following, we shall discuss how to reduce the cost by a recursive calculation algorithm.

**Definition 3**. Let $\lambda = [\{(i_1^{(1)},\cdots,i_d^{(1)}),\cdots,(i_1^{(r)},\cdots,i_d^{(r)})\}]$ and $\nu = [(j_1^{(1)},\cdots,j_d^{(1)}),\cdots,(j_1^{(r)},\cdots,j_d^{(r)})]$. $\lambda$ and $\nu$ are two different permutation equivalent subsets of the whole index set $U$. We say that the partition order of $\nu$ is less than that of $\lambda$, i.e., $\nu \prec \lambda$, if $\lambda$ can be converted to $\nu$ by merging two or more index elements. For instance, $\nu = [(1,1),(2,3)] \prec \lambda = [(1,2),(3,4)]$, since by merging 1 and 2, $\lambda$ is converted to $[\{(1, 1), (3, 4)\}] = [\{(1, 1), (2, 3)\}]$. $[\{(1, 1), (3, 4)\}]$ and $[\{(1, 1), (2, 3)\}]$ are the same permutation equivalent index subsets because we can apply the permutation $^{'}$: 1  1, 2  4, 3  3, 4  2 to $[\{(1, 1), (3, 4)\}]$. Note that the merging operation may not be unique, for example, $\nu$ can also be converted to $\lambda$ by merging 3 and 4. To clarify the concept of partition order, we list the order of all partitions when $d$=2 and $r$=2 in figure 1. The partition order of a permutation equivalent subset $\nu$ is said to be lower than that of another permutation equivalent subset $\lambda$ if there is a directed path from $\lambda$ to $\nu$.

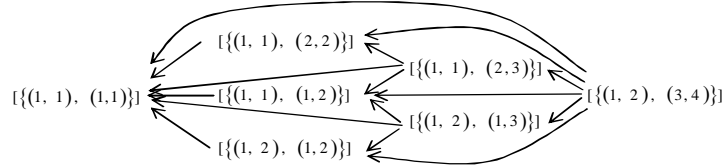

Figure 1: Order of all permutation equivalent subsets when $d = 2$ and r = $2$.

The difficulty for computing data partition sum and index partition sum comes from two constraints; equal constraint and unequal constraint. For example, in the permutation equivalent subset $[\{(1, 1), (2, 2)\}]$, the equal constraint is that the first and the second index number are equal and the third and fourth index are also equal. On the other hand, the unequal constraint requires that the first two index numbers are different from those of the last two. Due to the difficulties mentioned, we solve this problem by first relaxing the unequal constraint and then applying the principle of inclusion and exclusion. Thus, the calculation of a partition sum can be separated into two parts: the relaxed partition sum without unequal constraint, and lower order partition sums. For example,

$$w_{\lambda=[(1,1),(2,2)]} = \sum_{i \neq j}(w(i,i)w(j,j)) = w_{\lambda=[(1,1),(2,2)]^*} - w_{\lambda=[(1,1),(1,1)]} =$$

$$= \sum_{i,j}(w(i,i)w(j,j)) - \sum_{i=j}(w(i,i)w(j,j)) = (\sum_i w(i,i))^2 - \sum_i w(i,i)^2 ,$$ as the relaxed index partition

sum $w_{\lambda=[(1,1),(2,2)]^*} = \sum_{i,j}(w(i,i)w(j,j)) = (\sum_i w(i,i))^2 .$

**Proposition 3.** The index partition sum $w_\lambda$ can be calculated by subtracting all lower order partition sums from the corresponding relaxed index partition sum $w_\lambda^*$, i.e.,

$w_\lambda = w_\lambda^* - \sum_{v \prec \lambda} w_v \dfrac{\#(\lambda)}{\#(v)} \#(\lambda \to v)$, where $\#(\lambda)$ is the number of distinct order-sensitive permutation equivalent subsets. For example, there are 2!2!2!/2!/2!=2 order-sensitive index partition types for $\lambda = [(1,1),(2,3)]$. They are [(1, 1), (2, 3)] and [(2, 3), (1, 1)]. Note that [(1, 1), (2, 3)] and [(1, 1), (3, 2)] are the same type. $\#(\lambda \to v)$ is the number of different ways of merging a higher order permutation equivalent subset $\lambda$ to a low order permutation equivalent subset $v$.

The calculation of the data index partition sum is similar. Therefore, the computational cost mainly depends on the calculation of relaxed partition sum and the lowest order partition sum. Since the computational cost of the lowest order term is $O(n)$, we mainly discuss the calculation of relaxed partition sums in the following paragraphs.

To reduce the computational cost, we develop a greedy graph search algorithm. For demonstration, we use the following example.

$w^*_{\lambda=[(1,1),(1,2),(1,2),(1,3),(2,3),(1,4)]} = \#(\lambda) \sum\limits_{i,j,k,l} w(i,i)w(i,j)w(i,j)w(i,k)w(j,k)w(i,l)$. The permutation equivalent index subset is represented by an undirected graph. Every node denotes an index number. We connect two different nodes if these two corresponding index numbers are in the same index element, i.e., in the same small bracket. In figure 2, the number 2 on the edge $ij$ denotes that the pair $(i, j)$ is used twice. The self-connected node is also allowed. We assume there is no isolated subgraph in the following discussion. If any isolated subgraph exists, we only need to repeat the same procedure for all isolated subgraphs.

Now we shall discuss the steps to compute the $w^*_{\lambda=[(1,1),(1,2),(1,2),(1,3),(2,3),(1,4)]}$. Firstly, we get rid of the weights of edges and self-connections, i.e., $\sum\limits_{i,j,k,l} w(i,i)w(i,j)w(i,j)w(i,k)w(j,k)w(i,l)$

$= \sum\limits_{i,j,k,l} a(i,j)w(i,k)w(j,k)w(i,l)$, as $a(i,j) = w(i,i)w(i,j)w(i,j)$. Then we search a node with the lowest degree and do summation for all indices connected with respect to the chosen node, i.e., $\sum\limits_{i,j,k,l} a(i,j)w(i,k)w(j,k)w(i,l) = \sum\limits_{i,j,k} b(i,j)w(i,k)w(j,k)$, as $b(i,j) = \sum\limits_l a(i,j)w(i,l)$. The chosen nodes and connected edges are deleted after the above computation. We repeat the same step until a symmetric graph occurs. Since every node in the symmetric graph has the same degree, we randomly choose any node; for example, $k$ for summation, then $\sum\limits_{i,j,k} b(i,j)w(i,k)w(j,k) = \sum\limits_{i,j} b(i,j)c(i,j)$, as

$c(i,j) = \sum\limits_k w(i,k)w(j,k)$. Finally, we clear the whole graph and obtain the relaxed index partition sum.

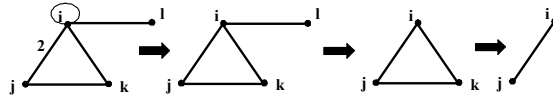

Figure 2: Greedy Search Algorithm for computing

The most computational-expensive case is the complete graph in which every pair of nodes is connected. Hence, the computational cost of $c_\lambda^*$ is determined by the subtotal that has the largest symmetric subgraph in its graph representation. For example, the most expensive relaxed index partition sum for $d=2$ and $r=3$ is $w(i,j)w(i,k)w(j,k)$, which is a triangle in the graph representation.

**Proposition 4** For $d>=2$, let $m(m-1)/2 \le r(d-1)d/2 < (m+1)m/2$, where $r$ is the order of moment and $m$ is an integer. For a $d$-th order test statistic, the computational cost of the partition sum for the $r$-th moment is bounded by $O(n^m)$. When $d = 1$ the computational complexity of the partition sum is $O(n)$.

Specifically, the computational cost of the 3rd and 4th moments for a second order test statistic is $O(n^3)$. The computational cost for the 1st and 2nd moments is $O(n^2)$.

## 2.4    Fitting

The Pearson distribution series (Pearson I ~ VII) is a family of probability distributions that are more general than the normal distribution [12]. It covers all distributions in the ($\beta1$, $\beta2$) plane including normal, beta, gamma, log-normal, and etc., where distribution shape parameters $\beta1$, $\beta2$ are the square of standardized skewness and kurtosis measurements, respectively. Given the first four moments, the Pearson distribution series can be utilized to approximate the permutation distribution of the test statistic without conducting real permutation.

## 3    Experimental results

To evaluate the accuracy and efficiency of our moments-based permutation tests, we generate simulated data and conduct permutation tests for both linear and quadratic test statistics. We consider six simulated cases in the first experiment for testing the difference between two groups, A and B. We use mean difference statistics here. For group A, $n_1$ observations are generated independently from Normal(0,1) in Cases 1-2, from Gamma(3,3) in Cases 3-4, and from Beta(0.8, 0.8) in Cases 5-6. For group B, $n_2$ independent observations are generated from Normal(1, 0.5) in Cases 1-2, from Gamma (3,2) in Cases 3-4, and from Beta(0.1, 0.1) in Cases 5-6. The design is balanced in Cases 1, 3, and 5 with $n_1 = n_2 = 10$, and unbalanced in Cases 2, 4, and 6 with $n_1 = 6$, $n_2 = 18$.

Table 1 illustrates the high accuracy of our moments-based permutation technique. Furthermore, comparing with exact permutation or random 10,000 permutations, the moments-based permutation tests reduce more than 99.8% of the computation cost, and this efficiency gain increases with sample size. Table 1 shows the computation time and $p$-values of three permutation methods from one simulation. In order to demonstrate the robustness of our method, we repeated the simulation for 10 times in each case, and calculated the mean and variance of the absolute biases of $p$-values of both moments-based permutation and random permutation, treating the $p$-values of exact permutation as gold standard. In most cases, our moments-based permutation is less biased and more stable than random permutation (Table 2), which demonstrates the robustness and accuracy of our method.

Table 1: Comparison of computation costs and $p$-values of three permutation methods: Moments-based permutation (MP), random permutation (RP), and exact permutation (EP). The $t\_MP$, $t\_RP$, and $t\_EP$ denote the computation time (in seconds), and $p\_MP$, $p\_RP$, and $p\_EP$ are the $p$-values of the three permutation methods.

|         | Case 1   | Case 2   | Case 3   | Case 4   | Case 5   | Case 6   |
|---------|----------|----------|----------|----------|----------|----------|
| $t\_MP$ | 6.79e-4  | 5.37e-4  | 5.54e-4  | 5.16e-4  | 5.79e-4  | 6.53e-4  |
| $t\_RP$ | 5.07e-1  | 5.15e-1  | 5.06e-1  | 1.30e-1  | 2.78e-1  | 5.99e-1  |
| $t\_EP$ | 3.99e-0  | 1.21e-0  | 3.71e-0  | 1.21e-0  | 3.71e-0  | 1.22e-0  |
| $p\_MP$ | 1.19e-1  | 2.45e-2  | 1.34e-1  | 1.19e-1  | 3.58e-2  | 5.07e-5  |
| $p\_RP$ | 1.21e-1  | 2.56e-2  | 1.36e-1  | 1.20e-1  | 3.53e-2  | 5.09e-2  |
| $p\_EP$ | 1.19e-1  | 2.39e-2  | 1.34e-1  | 1.15e-1  | 3.55e-2  | 5.11e-2  |

We consider three simulated cases in the second experiment for testing the difference among three groups D, E, and F. We use modified $F$ statistics [7] here. For group D, $n_1$ observations are generated independently from Normal(0,1) in Case 7, from Gamma(3,2) in Case 8, and from Beta(0.8, 0.8) in Case 9. For group E, $n_2$ independent observations are generated from Normal(0,1) in Case 7, from Gamma(3,2) in Case 8, and from Beta(0.8, 0.8) in Case 9. For group F, $n_3$ independent observations are generated from Normal(0.1,1) in Case 7, from Gamma(3,1) in Case 8, and from Beta(0.1, 0.1) in Case 9.The design is unbalanced with $n_1 = 6$, $n_2 = 8$, and $n_3 = 12$. Since the exact permutation is too expensive here, we consider the $p$-values of 200,000 random permutations (EP) as gold standard. Our methods are more than one hundred times faster than 2,000 random permutation (RP) and also more accurate and robust (Table 3).

We applied the method to the MRI hippocampi belonging to 2 groups, with 21 subjects in group A and 15 in group B. The surface shapes of different objects are represented by the same number of location vectors (with each location vector consisting of the spatial $x$, $y$, and $z$ coordinates of the corresponding vertex) for our subsequent statistical shape analysis. There is no shape difference at a location if the corresponding location vector has an equal

mean between two groups. Evaluation of the hypothesis test using our moments-based permutation with the modified Hotelling's $T^2$ test statistics [8] is shown in Fig. 3(a) and 3(b). It can be seen that the Pearson distribution approximation leads to ignorable discrepancy with the raw $p$-value map from real permutation. The false positive error control results are shown in Fig. 3(c).

Table 2: Robustness and accuracy comparison of moments-based permutation and random permutation across 10 simulations, considering the $p$-values of exact permutation as gold standard. Mean_ABias_MP and VAR_MP are the mean of the absolute biases and the variance of the biases of p-values of moments-based permutation; Mean_ABias_RP and VAR_RP are the mean of the absolute biases and the variance of the biases of $p$-values of random permutation. Mean difference statistic is used.

|  | Case1 | Case2 | Case3 | Case4 | Case5 | Case6 |
|---|---|---|---|---|---|---|
| Mean_ABias_MP | 1.62e-4 | 3.04e-4 | 6.36e-4 | 8.41e-4 | 1.30e-3 | 3.50e-3 |
| Mean_ABias_RP | 7.54e-4 | 3.39e-4 | 9.59e-4 | 8.39e-4 | 1.30e-3 | 2.00e-3 |
| VAR_MP | 6.42e-8 | 2.74e-7 | 1.54e-6 | 1.90e-6 | 3.76e-6 | 2.77e-5 |
| VAR_RP | 7.85e-7 | 1.86e-7 | 1.69e-6 | 3.03e-6 | 4.24e-5 | 1.88e-5 |

Table 3: Computation cost, robustness, and accuracy comparison of moments-based permutation and random permutation across 10 simulations. Modified $F$ statistic is used.

|  | Case7 | Case8 | Case9 |  | Case7 | Case8 | Case9 |
|---|---|---|---|---|---|---|---|
| t_MP | 1.03e-3 | 1.42e-3 | 1.64e-3 | Mean_ABias_MP | 9.23e-4 | 2.37e-4 | 2.11e-3 |
| t_RP | 1.51e-1 | 1.48e-1 | 1.38e-1 | Mean_ABias_RP | 3.94e-3 | 2.79e-3 | 3.42e-3 |
| t_EP | 1.76e+1 | 1.86e+1 | 2.37e+1 | VAR_MP | 1.10e-6 | 8.74e-8 | 1.23e-5 |
|  |  |  |  | VAR_RP | 2.27e-5 | 1.48e-5 | 1.85e-5 |

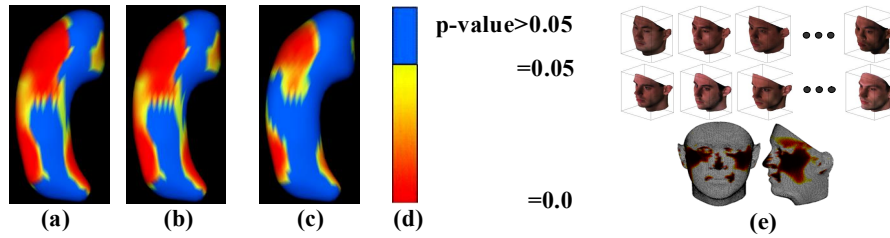

**(a)** **(b)** **(c)** **(d)** **(e)**

Figure 3. (a) and (b): Comparison of techniques in raw $p$-value measurement at $\alpha = 0.05$ (without correction), through real permutation ((a); number of permutations = 10,000) and using the present moments-based permutation (b). (c) $p$-map after BH's FDR correction of (b). (e) Facial differences between Asian male and white male. Locations in red on the 3D surface denote significant face shape differences (significance level $\alpha = 0.01$ with false discovery rate control).

We also applied our method to the 3D face comparison between Asian males and white males. We choose 10 Asian males and 10 white males out of the USF face database to calculate their differences with the modified Hotelling's $T^2$ test statistics. Each face surface is represented by 4,000 voxels. All surfaces are well aligned. Results from our algorithm in Fig. 3(e) show that significant differences occur at eye edge, nose, lip corners, and cheeks. They are consistent with anthropology findings and suggest the discriminant surface regions for ethnic group recognition.

## 4    Conclusion

We present and develop novel moments-based permutation tests where the permutation distributions are accurately approximated through Pearson distributions for considerably reduced computation cost. Comparing with regular random permutation, the proposed method considerably reduces computation cost without loss of accuracy. General and analytical formulations for the moments of permutation distribution are derived for weighted $v$-test statistics. The proposed strategy takes advantage of nonparametric permutation tests and parametric Pearson distribution approximation to achieve both accuracy/flexibility and efficiency.

## References

[1]    Nichols, T. E., and A. P. Holmes (2001), Nonparametric permutation tests for functional neuroimaging: A primer with examples, *Human Brain Mapping*, 15, 1-25.

[2]    Zhou, C., D. C. Park, M. Styner, and Y. M. Wang (2007), ROI constrained statistical surface morphometry, *IEEE International Symposium on Biomedical Imaging*, Washington, D. C., 1212-1215.

[3]    Zhou, C., and Y. M. Wang (2008), Hybrid permutation test with application to surface shape analysis, *Statistica Sinica*, 18, 1553-1568.

[4]    Pantazis, D., R. M. Leahy, T. E. Nichols, and M. Styner (2004), Statistical surface-based morphometry using a non-parametric approach, *IEEE International Symposium on Biomedical Imaging*, 2, 1283-1286.

[5]    Zhou, C., Y. Hu, Y. Fu., H. Wang, Y. M. Wang, and T. S. Huang (2008), 3D face analysis for distinct features using statistical randomization, *IEEE International Conference on Acoustics, Speech, and Signal Processing*, Las Vegas, Nevada, 981-984.

[6]    Hubert, L. (1987), *Assignment Methods in Combinatorial Data Analysis*, Marcel Dekker, New York.

[7]    Mielke, P. W., and K. J. Berry (2001), *Permutation Methods: A Distance Function Approach*, Springer, New York.

[8]    Good, P. (2005), Permutation, *Parametric and Bootstrap Tests of Hypotheses*, 3rd ed., Springer, New York.

[9]    Serfling, R. J. (1980), *Approximation Theorems of Mathematical Statistics*, Wiley, New York.

[10]   Edgington, E., and P. Onghena (2007), *Randomization Tests*, 4th ed., Chapman & Hall, London.

[11]   Nicholson, W. K. (2006), *Introduction to Abstract Algebra*, 3rd ed., Wiley, New York.

[12]   Hahn, G. J., and S. S. Shapiro (1967), *Statistical Models in Engineering*, John Wiley and Sons, Chichester, England.
